# Improving a Page Classifier with Anchor Extraction and Link Analysis

**William W. Cohen**
Center for Automated Learning and Discovery,
Carnegie-Mellon University
5000 Forbes Ave, Pittsburgh, PA 15213
william@wcohen.com

## Abstract

Most text categorization systems use simple models of documents and document collections. In this paper we describe a technique that improves a simple web page classifier's performance on pages from a new, unseen web site, by exploiting link structure within a site as well as page structure within hub pages. On real-world test cases, this technique significantly and substantially improves the accuracy of a bag-of-words classifier, reducing error rate by about half, on average. The system uses a variant of co-training to exploit unlabeled data from a new site. Pages are labeled using the base classifier; the results are used by a restricted wrapper-learner to propose potential "main-category anchor wrappers"; and finally, these wrappers are used as features by a third learner to find a categorization of the site that implies a simple hub structure, but which also largely agrees with the original bag-of-words classifier.

## 1 Introduction

Most text categorization systems use simple models of documents and document collections. For instance, it is common to model documents as "bags of words", and to model a collection as a set of documents drawn from some fixed distribution. An interesting question is how to exploit more detailed information about the structure of individual documents, or the structure of a collection of documents.

For web page categorization, a frequently-used approach is to use hyperlink information to improve classification accuracy (e.g., [7, 9, 15]). Often hyperlink structure is used to "smooth" the predictions of a learned classifier, so that documents that (say) are pointed to by the same "hub" page will be more likely to have the same classification after smoothing. This smoothing can be done either explicitly [15] or implicitly (for instance, by representing examples so that the distance between examples depends on hyperlink connectivity [7, 9]).

The structure of individual pages, as represented by HTML markup structure or linguis-

tic structure, is less commonly used in web page classification: however, page structure is often used in *extracting* information from web pages. Page structure seems to be particularly important in finding site-specific extraction rules ("wrappers"), since on a given site, formatting information is frequently an excellent indication of content [6, 10, 12].

This paper is based on two practical observations about web page classification. The first is that for many categories of economic interest (e.g., product pages, job-posting pages, and press releases) many sites contain "hub" or index pages that point to essentially all pages in that category on a site. These hubs rarely link *exclusively* to pages of a single category—instead the hubs will contain a number of additional links, such as links back to a home page and links to related hubs. However, the *page structure* of a hub page often gives strong indications of which links are to pages from the "main" category associated with the hub, and which are ancillary links that exist for other (e.g., navigational) purposes.

As an example, refer to Figure 1. Links to pages in the main category associated with this hub (previous NIPS conference homepages) are in the left-hand column of the table, and hence can be easily identified by the page structure.

The second observation is that it is relatively easy to learn to extract links from hub pages to main-category pages using existing *wrapper-learning* methods [8, 6]. Wrapper-learning techniques interactively learn to extract data of some type from a single site using user-provided training examples. Our experience in a number of domains indicates that main-category links on hub pages (like the NIPS-homepage links from Figure 1) can almost always be learned from two or three positive examples.

Exploiting these observations, we describe in this paper a web page categorization system that exploits link structure within a site, as well as page structure within hub pages, to improve classification accuracy of a traditional bag-of-words classifier on pages from a previously unseen site. The system uses a variant of co-training [3] to exploit unlabeled data from a new, previously unseen site. Specifically, pages are labeled using a simple bag-of-words classifier, and the results are used by a restricted wrapper-learner to propose potential "main-category link wrappers". These wrappers are then used as features by a decision tree learner to find a categorization of the pages on the site that implies a simple hub structure, but which also largely agrees with the original bag-of-words classifier.

## 2  One-step co-training and hyperlink structure

Consider a binary bag-of-words classifier $f$ that has been learned from some set of labeled web pages $D_\ell$. We wish to improve the performance of $f$ on pages from an unknown web site $S$, by smoothing its predictions in a way that is plausible given the hyperlink of $S$, and the page structure of potential hub pages in $S$. As background for the algorithm, let us consider first co-training, a well-studied approach for improving classifier performance using unlabeled data [3].

In *co-training* one assumes a concept learning problem where every instance $x$ can be written as a pair $(\mathbf{x_1}, \mathbf{x_2})$ such that $\mathbf{x_1}$ is conditionally independent of $\mathbf{x_2}$ given the class $y$. One also assumes that both $\mathbf{x_1}$ and $\mathbf{x_2}$ are sufficient for classification, in the sense that the target function $f(x)$ can be written either as a function of $\mathbf{x_1}$ or $\mathbf{x_2}$, i.e., that there exist functions $f_1(\mathbf{x_1}) = f(x)$ and $f_2(\mathbf{x_2}) = f(x)$. Finally one assumes that both $f_1$ and $f_2$ are learnable, i.e., that $f_1 \in H_1$ and $f_2 \in H_2$ and noise-tolerant learning algorithms $A_1$ and $A_2$ exist for $H_1$ and $H_2$.

⋮

**Webpages and Papers for Recent NIPS Conferences**

A. David Redish (dredish@cs.cmu.edu) created and maintained these web pages from 1994 until 1996. L. Douglas Baker (ldbapp+nips@cs.cmu.edu) maintained these web pages from 1997 until 1999. They were maintained in 2000 by L. Douglas Baker and Alexander Gray (agray+nips@cs.cmu.edu).

| | |
|---|---|
| NIPS*2000 | NIPS 13, the conference proceedings for 2000 ("Advances in Neural Information Processing Systems 13", edited by Leen, Todd K., Dietterich, Thomas G. and Tresp, Volker will be available to all attendees in June 2001. |
| | ∗ Abstracts and papers from this forthcoming volume are available on-line. |
| | ∗ BibTeX entries for all papers from this forthcoming volume are available on-line. |
| NIPS*99 | NIPS 12 is available from MIT Press. |
| | Abstracts and papers from this volume are available on-line. |
| NIPS*98 | NIPS 11 is available from MIT Press. |
| | Abstracts and (some) papers from this volume are available on-line. |

⋮

Figure 1: Part of a "hub" page. Links to pages in the main category associated with this hub are in the left-hand column of the table.

In this setting, a large amount of unlabeled data $D_u$ can be used to improve the accuracy of a small set of labeled data $D_\ell$, as follows. First, use $A_1$ to learn an approximation $f_1'$ to $f_1$ using $D_\ell$. Then, use $f_1'$ to label the examples in $D_u$, and use $A_2$ to learn from this training set. Given the assumptions above, $f_1'$'s errors on $D_u$ will appear to $A_2$ as random, uncorrelated noise, and $A_2$ can in principle learn an arbitrarily good approximation to $f$, given enough unlabeled data in $D_u$. We call this process *one-step co-training* using $A_1$, $A_2$, and $D_u$.

Now, consider a set $D_S$ of unlabeled pages from a unseen web site $S$. It seems not unreasonable to assume that the words $\mathbf{x_1}$ on a page $x \in S$ and the hub pages $\mathbf{x_2} \in S$ that hyperlink to $x$ are independent, given the class of $x$. This suggests that one-step co-training could be used to improve a learned bag-of-words classifier $f_1'$, using the following algorithm:

Algorithm 1 (One-step co-training):

1. *Parameters.* Let $S$ be a web site, $f_1'$ be a bag-of-words page classifier, and $D_S$ be the pages on the site $S$.

2. *Instance generation and labeling.* For each page $x^i \in D_S$, represent $x^i$ as a vector of all pages in $S$ that hyperlink to $x^i$. Call this vector $\mathbf{x}_2^i$. Let $y^i = f_1'(x_i)$.

3. *Learning.* Use a learner $A_2$ to learn $f_2'$ from the labeled examples $D_2 = \{(\mathbf{x}_2^i, y^i)\}_i$.

4. *Labeling.* Use $f_2'(x)$ as the final label for each page $x \in D_S$.

This "one-step" use of co-training is consistent with the theoretical results underlying co-training. In experimental studies, co-training is usually done iteratively, alternating between using $f_1'$ and $f_2'$ for tagging the unlabeled data. The one-step version seems more appropriate in this setting, in which there are a limited number of unlabeled examples over which each $\mathbf{x}_2$ is defined.

## 3 Anchor Extraction and Page Classification

### 3.1 Learning to extract anchors from web pages

Algorithm 1 has some shortcomings. Co-training assumes a *large* pool of unlabeled data: however, if the informative hubs for pages on $S$ are mostly within $S$ (a very plausible assumption) then the amount of useful unlabeled data is limited by the size of $S$. With limited amounts of unlabeled data, it is very important that $A_2$ has a strong (and appropriate) statistical bias, and that $A_2$ has some effective method for avoiding overfitting.

As suggested by Figure 1, the informativeness of hub features can be improved by using knowledge of the structure of hub pages themselves. To make use of hub page structure, we used a wrapper-learning system called $\text{WL}^2$, which has experimentally proven to be effective at learning substructures of web pages [6]. The output of $\text{WL}^2$ is an *extraction predicate*: a binary relation $p$ between pages $x$ and substrings $a$ within $x$. As an example, $\text{WL}^2$ might output $p = \{(x, a) : x$ is the page of Figure 1 and $a$ is an anchor appearing in the first column of the table$\}$. (An *anchor* is a substring of a web page that defines a hyperlink.)

This suggests a modification of Algorithm 1, in which one-step co-training is carried out on the problem of extracting anchors rather than the problem of labeling web pages. Specifically, one might map $f_1$'s predictions from web pages to anchors, by giving a positive label to anchor $a$ iff $a$ links to a page $x$ such that $f_1'(x) = 1$; then use $\text{WL}^2$ algorithm $A_2$ to learn a predicate $p_2'$; and finally, map the predictions of $p_2'$ from anchors back to web pages.

One problem with this approach is that $\text{WL}^2$ was designed for user-provided data sets, which are small and noise-free. Another problem is that it unclear how to map class labels from anchors back to web pages, since a page might be pointed to by many different anchors.

### 3.2 Bridging the gap between anchors and pages

Based on these observations we modified Algorithm 1 as follows. As suggested, we map the predictions about page labels made by $f_1'$ to anchors. Using these anchor labels, we then produce many *small* training sets that are passed to $\text{WL}^2$. The intuition here is that *some* of these training sets will be noise-free, and hence similar to those that might be provided by a user. Finally, we use the many wrappers produced by $\text{WL}^2$ as features in a representation of a page $x$, and again use a learner to combine the wrapper-features and produce a single classification for a page.

Algorithm 2:

1. *Parameters*. Let $S$ be a web site, $f_1'$ be a bag-of-words page classifier, and $D_S$ be the pages on the site.

2. *Link labeling*. For each anchor $a$ on a page $x \in S$, label $a$ as *tentatively-positive*

if $a$ points to a page $x'$ such that $x' \in S$ and $f_1'(x') = 1$.

3. *Wrapper proposal*. Let $P$ be the set of all pairs $(x, a)$ where $a$ is a tentatively-positive link and $x$ is the page on which $a$ is found. Generate a number of small sets $D_1, \ldots, D_k$ containing such pairs, and for each subset $D_i$, use WL$^2$ to produce a number of possible extraction predicates $p_{i,1}, \ldots, p_{i,k_i}$. (See appendix for details).

4. *Instance generation and labeling*. We will say that the "wrapper predicate" $p_{ij}$ *links to* $x$ iff $p_{ij}$ includes some pair $(x', a)$ such that $x' \in D_S$ and $a$ is a hyperlink to page $x$. For each page $x^i \in D_S$, represent $x^i$ as a vector of all wrappers $p_{ij}$ that link to $x$. Call this vector $\mathbf{x}_2^i$. Let $y^i = f_1'(x_i)$.

5. *Learning*. Use a learner $A_2$ to learn $f_2'$ from the labeled examples $D_S = \{(\mathbf{x}_2^i, y^i)\}_i$.

6. *Labeling*. Use $f_2'(x)$ as the final label for each page $x \in D_S$.

A general problem in building learning systems for new problems is exploiting existing knowledge about these problems. In this case, in building a page classifier, one would like to exploit knowledge about the related problem of link extraction. Unfortunately this knowledge is not in any particularly convenient form (e.g., a set of well-founded parametric assumptions about the data): instead, we only know that experimentally, a certain learning algorithm works well on the problem. In general, it is often the case that this sort of experimental evidence is available, even when a learning problem is not formally well-understood.

The advantage of Algorithm 2 is that one need make no parametric assumptions about the anchor-extraction problem. The bagging-like approach of "feeding" WL$^2$ many small training sets, and the use of a second learning algorithm to aggregate the results of WL$^2$, are a means of exploiting prior experimental results, in lieu of more precise statistical assumptions.

## 4    Experimental results

To evaluate the technique, we used the task of categorizing web pages from company sites as *executive biography* or *other*. We selected nine company web sites with non-trivial hub structures. These were crawled using a heuristic spidering strategy intended to find executive biography pages with high recall.[1] The crawl found 879 pages, of which 128 were labeled positive. A simple bag-of-words classifier $f_1'$ was trained using a disjoint set of sites (different from the nine above), obtaining an average accuracy of 91.6% (recall 82.0%, precision 61.8%) on the nine held-out sites. Using an implemention of Winnow [2, 11] as $A_2$, Algorithm 2 obtained an average accuracy of 96.4% on the nine held-out sites. Algorithm 2 improves over the baseline classifier $f_1'$ on six of the nine sites, and obtains the same accuracy on two more. This difference is significant at the 98% level with a 2-tailed paired sign test, and at the 95% level with a 2-tailed paired $t$ test.

Similar results were also obtained using a sparse-feature implementation of a C4.5-like decision tree learning algorithm [14] for learner $A_2$. (Note that both Winnow and C4.5 are known to work well when data is noisy, irrelevant attributes are present, and the underlying concept is "simple".) These results are summarized in Table 1.

| Site | Classifier $f_1'$ | | Algorithm 2 (C4.5) | | Algorithm 2 (Winnow) | |
|------|----------|--------|----------|--------|----------|--------|
|      | Accuracy | (SE)   | Accuracy | (SE)   | Accuracy | (SE)   |
| 1    | 1.000    | (0.000) | 0.960   | (0.028) | 0.960   | (0.028) |
| 2    | 0.932    | (0.027) | 0.955   | (0.022) | 0.955   | (0.022) |
| 3    | 0.813    | (0.028) | 0.934   | (0.018) | 0.939   | (0.017) |
| 4    | 0.904    | (0.029) | 0.962   | (0.019) | 0.962   | (0.019) |
| 5    | 0.939    | (0.024) | 0.960   | (0.020) | 0.960   | (0.020) |
| 6    | 1.000    | (0.000) | 1.000   | (0.000) | 1.000   | (0.000) |
| 7    | 0.918    | (0.028) | 0.990   | (0.010) | 0.990   | (0.010) |
| 8    | 0.788    | (0.044) | 0.882   | (0.035) | 0.929   | (0.028) |
| 9    | 0.948    | (0.029) | 0.948   | (0.029) | 0.983   | (0.017) |
| avg  | 0.916    |        | 0.954   |        | 0.964   |        |

Table 1: Experimental results with Algorithm 2. Paired tests indicate that both versions of Algorithm 2 significantly improve on the baseline classifier.

# 5 Related work

The introduction discusses the relationship between this work and a number of previous techniques for using hyperlink structure in web page classification [7, 9, 15]. The $WL^2$-based method for finding document structure has antecedents in other techniques for learning [10, 12] and automatically detecting [4, 5] structure in web pages.

In concurrent work, Blei et al [1] introduce a probabilistic model called "scoped learning" which gives a generative model for the situation described here: collections of examples in which some subsets (documents from the same site) share common "local" features, and all documents share common "content" features. Blei et al do not address the specific problem considered here, of using both page structure and hyperlink structure in web page classification. However, they do apply their technique to two closely related problems: they augment a page classification method with local features based on the page's URL, and also augment content-based classification of "text nodes" (specific substrings of a web page) with page-structure-based local features.

We note that Algorithm 2 could be adapted to operate in Blei et al's setting: specifically, the $x_2$ vectors produced in Steps 2-4 could be viewed as "local features". (In fact, Blei et al generated page-structure-based features for their extraction task in exactly this way: the only difference is that $WL^2$ was parameterized differently.) The co-training framework adopted here clearly makes different assumptions than those adopted by Blei et al. More experimentation is needed to determine which is preferable—current experimental evidence [13] is ambiguous as to when probabilistic approaches should be prefered to co-training.

# 6 Conclusions

We have described a technique that improves a simple web page classifier by exploiting link structure within a site, as well as page structure within hub pages. The system uses a variant of co-training called "one-step co-training" to exploit unlabeled data from a new site. First, pages are labeled using the base classifier. Next, results of this labeling are propogated to links to labeled pages, and these labeled links are used by a wrapper-learner called $WL^2$ to propose potential "main-category link wrappers". Finally, these wrappers

are used as features by another learner $A_2$ to find a categorization of the site that implies a simple hub structure, but which also largely agrees with the original bag-of-words classifier. Experiments suggest the choice of $A_2$ is not critical.

On a real-world benchmark problem, this technique substantially improved the accuracy of a simple bag-of-words classifier, reducing error rate by about half. This improvement is statistically significant.

### Acknowledgments

The author wishes to thank his former colleagues at Whizbang Labs for many helpful discussions and useful advice.

## Appendix A: Details on "Wrapper Proposal"

Extraction predicates are constructed by $\mathrm{WL}^2$ using a rule-learning algorithm and a configurable set of components called *builders*. Each builder $B$ corresponds to a language $\mathcal{L}_B$ of *extraction predicates*. Builders support a certain set of operations relative to $\mathcal{L}_B$, in particular, the *least general generalization (LGG) operation*. Given a set of pairs $D = \{(x_i, a_i)\}$ such that each $a_i$ is a substring of $x_i$, $LGG_B(D)$ is the *least general* $p \in \mathcal{L}_B$ such that $(x, a) \in D \Rightarrow (x, a) \in p$. Intuitively, $LGG_B(D)$ encodes common properties of the (positive) examples in $D$. Depending on $B$, these properties might be membership in a particular syntactic HTML structure (e.g., a specific table column), common visual properties (e.g., being rendered in boldface), etc.

To generate subsets $D_i$ in Step 3 of Algorithm 2, we used every pair of links that pointed to the two most confidently labeled examples; every pair of adjacent tentatively-positive links; and every triple and every quadruple of tentatively-positive links that were separated by at most 10 intervening tokens. These heuristics were based on the observation that in most extraction tasks, the items to be extracted are close together. Careful implementation allows the subsets $D_i$ to be generated in time linear in the size of the site. (We also note that these heuristics were initially developed to support a different set of experiments [1], and were not substantially modified for the experiments in this paper.)

Normally, $\mathrm{WL}^2$ is parameterized by a list $\mathcal{B}$ of builders, which are called by a "master" rule-learning algorithm. In our use of $\mathrm{WL}^2$, we simply applied each builder $B_j$ to a dataset $D_i$, to get the set of predicates $\{p_{ij}\} = \{LGG_{B_j}(D_i)\}$, instead of running the full $\mathrm{WL}^2$ learning algorithm.

## Footnotes

[1]The authors wish to thank Vijay Boyaparti for assembling this data set.

## References

[1] David M. Blei, J. Andrew Bagnell, and Andrew K. McCallum. Learning with scope, with application to information extraction and classification. In *Proceedings of UAI-2002*, Edmonton, Alberta, 2002.

[2] Avrim Blum. Learning boolean functions in an infinite attribute space. *Machine Learning*, 9(4):373–386, 1992.

[3] Avrin Blum and Tom Mitchell. Combining labeled and unlabeled data with co-training. In *Proceedings of the 1998 Conference on Computational Learning Theory*, Madison, WI, 1998.

[4] William W. Cohen. Automatically extracting features for concept learning from the web. In *Machine Learning: Proceedings of the Seventeeth International Conference*, Palo Alto, California, 2000. Morgan Kaufmann.

[5] William W. Cohen and Wei Fan. Learning page-independent heuristics for extracting data from web pages. In *Proceedings of The Eigth International World Wide Web Conference (WWW-99)*, Toronto, 1999.

[6] William W. Cohen, Lee S. Jensen, and Matthew Hurst. A flexible learning system for wrapping tables and lists in HTML documents. In *Proceedings of The Eleventh International World Wide Web Conference (WWW-2002)*, Honolulu, Hawaii, 2002.

[7] David Cohn and Thomas Hofmann. The missing link - a probabilistic model of document content and hypertext connectivity. In *Advances in Neural Information Processing Systems 13*. MIT Press, 2001.

[8] Lee S. Jensen and William W. Cohen. A structured wrapper induction system for extracting information from semi-structured documents. In *Proceedings of the IJCAI-2001 Workshop on Adaptive Text Extraction and Mining*, Seattle, WA, 2001.

[9] T. Joachims, N. Cristianini, and J. Shawe-Taylor. Composite kernels for hypertext categorisation. In *Proceedings of the International Conference on Machine Learning (ICML-2001)*, 2001.

[10] N. Kushmeric. Wrapper induction: efficiency and expressiveness. *Artificial Intelligence*, 118:15–68, 2000.

[11] Nick Littlestone. Learning quickly when irrelevant attributes abound: A new linear-threshold algorithm. *Machine Learning*, 2(4), 1988.

[12] Ion Muslea, Steven Minton, and Craig Knoblock. Wrapper induction for semistructured information sources. *Journal of Autonomous Agents and Multi-Agent Systems*, 16(12), 1999.

[13] Kamal Nigam and Rayyid Ghani. Analyzing the effectiveness and applicability of co-training. In *Proceedings of the Ninth International Conference on Information and Knowledge Management (CIKM-2000)*, 2000.

[14] J. Ross Quinlan. *C4.5: programs for machine learning*. Morgan Kaufmann, 1994.

[15] S. Slattery and T. Mitchell. Discovering test set regularities in relational domains. In *Proceedings of the 17th International Conference on Machine Learning (ICML-2000)*, June 2000.
